# Clustering with a Domain-Specific Distance Measure

**Steven Gold, Eric Mjolsness and Anand Rangarajan**
Department of Computer Science
Yale University
New Haven, CT 06520-8285

## Abstract

With a point matching distance measure which is invariant under translation, rotation and permutation, we learn 2-D point-set objects, by clustering noisy point-set images. Unlike traditional clustering methods which use distance measures that operate on feature vectors – a representation common to most problem domains – this object-based clustering technique employs a distance measure specific to a type of object within a problem domain. Formulating the clustering problem as two nested objective functions, we derive optimization dynamics similar to the Expectation-Maximization algorithm used in mixture models.

## 1 Introduction

Clustering and related unsupervised learning techniques such as competitive learning and self-organizing maps have traditionally relied on measures of distance, like Euclidean or Mahalanobis distance, which are generic across most problem domains. Consequently, when working in complex domains like vision, extensive preprocessing is required to produce feature sets which reflect properties critical to the domain, such as invariance to translation and rotation. Not only does such preprocessing increase the architectural complexity of these systems but it may fail to preserve some properties inherent in the domain. For example in vision, while Fourier decomposition may be adequate to handle reconstructions invariant under translation and rotation, it is unlikely that distortion invariance will be as amenable to this technique (von der Malsburg, 1988).

These problems may be avoided with the help of more powerful, domain-specific distance measures, including some which have been applied successfully to visual recognition tasks (Simard, Le Cun, and Denker, 1993; Huttenlocher *et al.*, 1993). Such measures can contain domain critical properties; for example, the distance measure used here to cluster 2-D point images is invariant under translation, rotation and labeling permutation. Moreover, new distance measures may constructed, as this was, using Bayesian inference on a model of the visual domain given by a probabilistic grammar (Mjolsness, 1992). Distortion invariant or graph matching measures, so formulated, can then be applied to other domains which may not be amenable to description in terms of features.

Objective functions can describe the distance measures constructed from a probabilistic grammar, as well as learning problems that use them. The clustering problem in the present paper is formulated as two nested objective functions: the inner objective computes the distance measures and the outer objective computes the cluster centers and cluster memberships. A clocked objective function is used, with separate optimizations occurring in distinct clock phases (Mjolsness and Miranker, 1993). The optimization is carried out with coordinate ascent/descent and deterministic annealing and the resulting dynamics is a generalization of the Expectation-Maximization (EM) algorithm commonly used in mixture models.

## 2 Theory

### 2.1 The Distance Measure

Our distance measure quantifies the degree of similarity between two unlabeled 2-D point images, irrespective of their position and orientation. It is calculated with an objective that can be used in an image registration problem. Given two sets of points $\{X_j\}$ and $\{Y_k\}$, one can minimize the following objective to find the translation, rotation and permutation which best maps $Y$ onto $X$ :

$$E_{reg}(m, t, \Theta) = \sum_{jk} m_{jk} \|X_j - t - R(\Theta) \cdot Y_k\|^2$$

with constraints: $\forall j \sum_k m_{jk} = 1$ , $\forall k \sum_j m_{jk} = 1$.

Such a registration permits the matching of two sparse feature images in the presence of noise (Lu and Mjolsness, 1994). In the above objective, $m$ is a permutation matrix which matches one point in one image with a corresponding point in the other image. The constraints on $m$ ensure that each point in each image corresponds to one and only one point in the other image (though note later remarks regarding fuzziness).

Then given two sets of points $\{X_j\}$ and $\{Y_k\}$ the distance between them is defined as:

$$D(\{X_j\}, \{Y_k\}) = \min_{m, t, \Theta} (E_{reg}(m, t, \Theta) \mid \text{constraints on } m) \ . \tag{1}$$

This measure is an example of a more general image distance measure derived in (Mjolsness, 1992):

$$d(x, y) = \min_T d(x, T(y)) \in [0, \infty)$$

where $T$ is a set of transformation parameters introduced by a visual grammar. In (1) translation, rotation and permutation are the transformations, however scaling

or distortion could also have been included, with consequent changes in the objective function.

The constraints are enforced by applying the Potts glass mean field theory approximations (Peterson and Soderberg,1989) and then using an equivalent form of the resulting objective, which employs Lagrange multipliers and an $x \log x$ barrier function (as in Yuille and Kosowsky, 1991):

$$E_{reg}(m,t,\Theta) \;=\; \sum_{jk} m_{jk}\|X_j - t - R(\Theta)\cdot Y_k\|^2 + \frac{1}{\beta}\sum_{jk} m_{jk}(\log m_{jk} - 1)$$

$$+ \sum_{j}\mu_j(\sum_{k} m_{jk} - 1) + \sum_{k}\nu_k(\sum_{j} m_{jk} - 1) \;. \tag{2}$$

In this objective we are looking for a saddle point. (2) is minimized with respect to $m$, $t$, and $\Theta$, which are the correspondence matrix, translation,and rotation, and is maximized with respect to $\mu$ and $\nu$, the Lagrange multipliers that enforce the row and column constraints for $m$.

## 2.2   The Clustering Objective

The learning problem is formulated as follows: Given a set of $I$ images, $\{X_i\}$, with each image consisting of $J$ points, find a set of $A$ cluster centers $\{Y_a\}$ and match variables $\{M_{ia}\}$ defined as

$$M_{ia} = \begin{cases} 1 & \text{if } X_i \text{ is in } Y_a\text{'s cluster} \\ 0 & \text{otherwise,} \end{cases}$$

such that each image is in only one cluster, and the total distance of all the images from their respective cluster centers is minimized. To find $\{Y_a\}$ and $\{M_{ia}\}$ minimize the cost function,

$$E_{cluster}(Y,M) = \sum_{ia} M_{ia} D(X_i, Y_a) \;,$$

with the constraint that $\forall i \; \sum_{a} M_{ia} = 1$. $D(X_i, Y_a)$, the distance function, is defined by (1).

The constraints on $M$ are enforced in a manner similar to that described for the distance measure, except that now only the rows of the matrix $M$ need to add to one, instead of both the rows and the columns. The Potts glass mean field theory method is applied and an equivalent form of the resulting objective is used:

$$E_{cluster}(Y,M) = \sum_{ia} M_{ia} D(X_i, Y_a) + \frac{1}{\beta}\sum_{ia} M_{ia}(\log M_{ia} - 1) + \sum_{i}\lambda_i(\sum_{a} M_{ia} - 1) \;.$$

$$\tag{3}$$

Replacing the distance measure by (2), we derive:

$$E_{cluster}(Y,M,t,\Theta,m) = \sum_{ia} M_{ia} \sum_{jk} m_{iajk}\|X_{ij} - t_{ia} - R(\Theta_{ia})\cdot Y_{ak}\|^2 +$$

$$\sum_{ia}[\frac{1}{\beta_m}\sum_{jk} m_{iajk}(\log m_{iajk} - 1) + \sum_{j}\mu_{iaj}(\sum_{k} m_{iajk} - 1) +$$

$$\sum_{k}\nu_{iak}(\sum_{j} m_{iajk} - 1)] + \frac{1}{\beta_M}\sum_{ia} M_{ia}(\log M_{ia} - 1) + \sum_{i}\lambda_i(\sum_{a} M_{ia} - 1) \;.$$

A saddle point is required. The objective is minimized with respect to $Y$, $M$, $m$, $t$, $\Theta$, which are respectively the cluster centers, the cluster membership matrix, the correspondence matrices, the rotations, and the translations. It is maximized with respect to $\lambda$, which enforces the row constraint for $M$, and $\mu$ and $\nu$ which enforce the column and row constraints for $m$. $M$ is a cluster membership matrix indicating for each image $i$, which cluster $a$ it falls within, and $m_{ia}$ is a permutation matrix which assigns to each point in cluster center $Y_a$ a corresponding point in image $X_i$. $\Theta_{ia}$ gives the rotation between image $i$ and cluster center $a$. Both $M$ and $m$ are fuzzy, so a given image may partially fall within several clusters, with the degree of fuzziness depending upon $\beta_m$ and $\beta_M$.

Therefore, given a set of images, $X$, we construct $E_{cluster}$ and upon finding the appropriate saddle point of that objective, we will have $Y$, their cluster centers, and $M$, their cluster memberships.

## 3   The Algorithm

### 3.1   Overview - A Clocked Objective Function

The algorithm to minimize the above objective consists of two loops - an inner loop to minimize the distance measure objective (2) and an outer loop to minimize the clustering objective (3). Using coordinate descent in the outer loop results in dynamics similar to the EM algorithm for clustering (Hathaway, 1986). (The EM algorithm has been similarly used in supervised learning [Jordan and Jacobs, 1993].) All variables occurring in the distance measure objective are held fixed during this phase. The inner loop uses coordinate ascent/descent which results in repeated row and column projections for $m$. The minimization of $m$, $t$ and $\Theta$ occurs in an incremental fashion, that is their values are saved after each inner loop call from within the outer loop and are then used as initial values for the next call to the inner loop. This tracking of the values of $m$, $t$, and $\Theta$ in the inner loop is essential to the efficiency of the algorithm since it greatly speeds up each inner loop optimization. Each coordinate ascent/descent phase can be computed analytically, further speeding up the algorithm. Local minima are avoided, by deterministic annealing in both the outer and inner loops.

The resulting dynamics can be concisely expressed by formulating the objective as a clocked objective function, which is optimized over distinct sets of variables in phases,

$$E_{clocked} = E_{cluster}\langle\langle\langle(\mu, m)^A, (\nu, m)^A\rangle_\oplus, \Theta^A, t^A\rangle_\oplus, (\lambda, M)^A, Y^A\rangle_\oplus$$

with this special notation employed recursively:

$E\langle x, y\rangle_\oplus$ : coordinate descent on $x$, then $y$, iterated (if necessary)
$x^A$ : use analytic solution for $x$ phase

The algorithm can be expressed less concisely in English, as follows:

Initialize $t$, $\Theta$ to zero, $Y$ to random values
**Begin** Outer Loop
  **Begin** Inner Loop
    Initialize $t$, $\Theta$ with previous values

Find $m$, $t$, $\Theta$ for each $ia$ pair :
    Find $m$ by softmax, projecting across $j$, then $k$, iteratively
    Find $\Theta$ by coordinate descent
    Find $t$ by coordinate descent
**End** Inner Loop
If first time through outer loop $\uparrow$ $\beta_m$ and repeat inner loop
Find $M$,$Y$ using fixed values of $m$, $t$, $\Theta$ determined in inner loop:
    Find $M$ by softmax, across $i$
    Find $Y$ by coordinate descent
$\uparrow$ $\beta_M$, $\beta_m$
**End** Outer Loop

When the distances are calculated for all the $X$ - $Y$ pairs the first time time through the outer loop, annealing is needed to minimize the objectives accurately. However on each succeeding iteration, since good initial estimates are available for $t$ and $\Theta$ (the values from the previous iteration of the outer loop) annealing is unnecessary and the minimization is much faster.

The speed of the above algorithm is increased by not recalculating the $X$ - $Y$ distance for a given $ia$ pair when its $M_{ia}$ membership variable drops below a threshold.

### 3.2 Inner Loop

The inner loop proceeds in three phases. In phase one, while $t$ and $\Theta$ are held fixed, $m$ is initialized with the softmax function and then iteratively projected across its rows and columns until the procedure converges. In phases two and three, $t$ and $\Theta$ are updated using coordinate descent. Then $\beta_m$ is increased and the loop repeats.

In phase one $m$ is updated with softmax:

$$m_{iajk} = \frac{\exp(-\beta_m \|X_{ij} - t_{ia} - R(\Theta_{ia}) \cdot Y_{ak}\|^2)}{\sum_{k'} \exp(-\beta_m \|X_{ij} - t_{ia} - R(\Theta_{ia}) \cdot Y_{ak'}\|^2)} \; .$$

Then $m$ is iteratively normalized across $j$ and $k$ until $\sum_{jk} \Delta m_{iajk} < \epsilon$ :

$$m_{iajk} = \frac{m_{iajk}}{\sum_{j'} m_{iaj'k}} \quad ; \quad m_{iajk} = \frac{m_{iajk}}{\sum_{k'} m_{iajk'}}$$

Using coordinate descent $\Theta$ is calculated in phase two:

$$\Theta_{ia} = \tan^{-1} \frac{\sum_{jk} m_{iajk}((X_{ij2}-t_{ia2})Y_{ak1}-(X_{ij1}-t_{ia1})Y_{ak2})}{\sum_{jk} m_{iajk}((X_{ij1}-t_{ia1})Y_{ak1}-(X_{ij2}-t_{ia2})Y_{ak2})}$$

And $t$ in phase three:

$$t_{ia1} = \frac{\sum_{jk} m_{iajk}(X_{ij1}-t_{ia1}-Y_{ak1}\cos\Theta_{ia}+Y_{ak2}\sin\Theta_{ia})}{\sum_{jk} m_{iajk}}$$

$$t_{ia2} = \frac{\sum_{jk} m_{iajk}(X_{ij2}-t_{ia2}-Y_{ak1}\sin\Theta_{ia}-Y_{ak2}\cos\Theta_{ia})}{\sum_{jk} m_{iajk}}$$

Finally $\beta_m$ is increased and the loop repeats.

By setting the partial derivatives of (2) to zero and initializing $\mu_j^0$ and $\nu_k^0$ to zero, the algorithm for phase one may be derived. Phases two and three may be derived by taking the partial derivative of (2) with respect to $\Theta$, setting it to zero, solving for $\Theta$, and then solving for the fixed point of the vector $(t_1, t_2)$.

Beginning with a small $\beta_m$ allows minimization over a fuzzy correspondence matrix $m$, for which a global minimum is easier to find. Raising $\beta_m$ drives the $m$'s closer to 0 or 1, as the algorithm approaches a saddle point.

### 3.3 Outer Loop

The outer loop also proceeds in three phases: (1) distances are calculated by calling the inner loop, (2) $M$ is projected across $a$ using the softmax function, (3) coordinate descent is used to update $Y$.

Therefore, using softmax $M$ is updated in phase two:

$$M_{ia} = \frac{\exp(-\beta_M \sum_{jk} m_{iajk} \|X_{ij} - t_{ia} - R(\Theta_{ia}) \cdot Y_{ak}\|^2)}{\sum_{a'} \exp(-\beta_M \sum_{jk} m_{ia'jk} \|X_{ij} - t_{ia'} - R(\Theta_{ia'}) \cdot Y_{a'k}\|^2)} .$$

$Y$, in phase three is calculated using coordinate descent:

$$Y_{ak1} = \frac{\sum_i M_{ia} \sum_j m_{iajk}(\cos \Theta_{ia}(X_{ij1} - t_{ia1}) + \sin \Theta_{ia}(X_{ij2} - t_{ia2}))}{\sum_i M_{ia} \sum_j m_{iajk}}$$

$$Y_{ak2} = \frac{\sum_i M_{ia} \sum_j m_{iajk}(-\sin \Theta_{ia}(X_{ij1} - t_{ia1}) + \cos \Theta_{ia}(X_{ij2} - t_{ia2}))}{\sum_i M_{ia} \sum_j m_{iajk}}$$

Then $\beta_M$ is increased and the loop repeats.

## 4    Methods and Experimental Results

In two experiments (Figures 1a and 1b) 16 and 100 randomly generated images of 15 and 20 points each are clustered into 4 and 10 clusters, respectively.

A stochastic model, formulated with essentially the same visual grammar used to derive the clustering algorithm (Mjolsness, 1992), generated the experimental data. That model begins with the cluster centers and then applies probabilistic transformations according to the rules laid out in the grammar to produce the images. These transformations are then inverted to recover cluster centers from a starting set of images. Therefore, to test the algorithm, the same transformations are applied to produce a set of images, and then the algorithm is run in order to see if it can recover the set of cluster centers, from which the images were produced.

First, $n = 10$ points are selected using a uniform distribution across a normalized square. For each of the $n = 10$ points a model prototype (cluster center) is created by generating a set of $k = 20$ points uniformly distributed across a normalized square centered at each orginal point. Then, $m = 10$ new images consisting of $k = 20$ points each are generated from each model prototype by displacing all $k$ model points by a random global translation, rotating all $k$ points by a random global rotation within a 54° arc, and then adding independent noise to each of the translated and rotated points with a Gaussian distribution of variance $\sigma^2$.

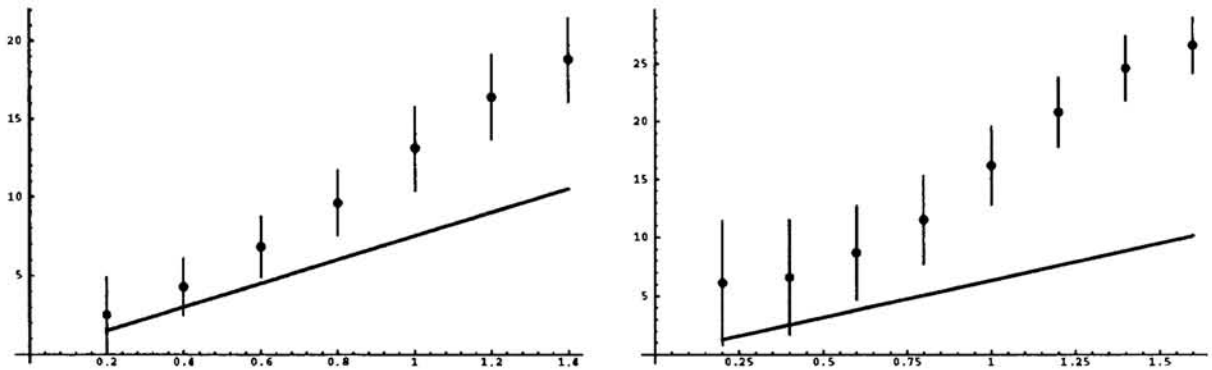

Figure 1: (a): 16 images, 15 points each (b):100 images, 20 points each

The $p = n \times m = 100$ images so generated is the input to the algorithm. The algorithm, which is initially ignorant of cluster membership information, computes $n = 10$ cluster centers as well as $n \times p = 1000$ match variables determining the cluster membership of each point image. $\sigma$ is varied and for each $\sigma$ the average distance of the computed cluster centers to the theoretical cluster centers (i.e. the original $n = 10$ model prototypes) is plotted.

Data (Figure 1a) is generated with 20 random seeds with constants of $n = 4, k = 15, m = 4, p = 16$, varying $\sigma$ from .02 to .14 by increments of .02 for each seed. This produces 80 model prototype-computed cluster center distances for each value of $\sigma$ which are then averaged and plotted, along with an error bar representing the standard deviation of each set. 15 random seeds (Figure 1b) with constants of $n = 10, k = 20, m = 10, p = 100$, $\sigma$ varied from .02 to .16 by increments of .02 for each seed, produce 150 model prototype-computed cluster center distances for each value of $\sigma$. The straight line plotted on each graph shows the expected model prototype-cluster center distances, $\hat{D} = k\sigma/\sqrt{n}$, which would be obtained if there were no translation or rotation for each generated image, and if the cluster memberships were known. It can be considered a lower bound for the reconstruction performance of our algorithm. Figures 1a and 1b together summarize the results of 280 separate clustering experiments.

For each set of images the algorithm was run four times, varying the initial randomly selected starting cluster centers each time and then selecting the run with the lowest energy for the results. The annealing rate for $\beta_M$ and $\beta_m$ was a constant factor of 1.031. Each run of the algorithm averaged ten minutes on an Indigo SGI workstation for the 16 image test, and four hours for the 100 image test. The running time of the algorithm is $O(pnk^2)$. Parallelization, as well as hierarchical and attentional mechanisms, all currently under investigation, can reduce these times.

## 5   Summary

By incorporating a domain-specific distance measure instead of the typical generic distance measures, the new method of unsupervised learning substantially reduces the amount of ad-hoc pre-processing required in conventional techniques. Critical features of a domain (such as invariance under translation, rotation, and permu-

tation) are captured within the clustering procedure, rather than reflected in the properties of feature sets created prior to clustering. The distance measure and learning problem are formally described as nested objective functions. We derive an efficient algorithm by using optimization techniques that allow us to divide up the objective function into parts which may be minimized in distinct phases. The algorithm has accurately recreated 10 prototypes from a randomly generated sample database of 100 images consisting of 20 points each in 120 experiments. Finally, by incorporating permutation invariance in our distance measure, we have a technique that we may be able to apply to the clustering of graphs. Our goal is to develop measures which will enable the learning of objects with shape or structure.

## Acknowledgements

This work has been supported by AFOSR grant F49620-92-J-0465 and ONR/DARPA grant N00014-92-J-4048.

## References

R. Hathaway. (1986) Another interpretation of the EM algorithm for mixture distributions. *Statistics and Probability Letters* **4**:53:56.

D. Huttenlocher, G. Klanderman and W. Rucklidge. (1993) Comparing images using the Hausdorff Distance. *Pattern Analysis and Machine Intelligence* **15**(9):850:863.

A. L. Yuille and J.J. Kosowsky. (1992). Statistical physics algorithms that converge. Technical Report 92-7, Harvard Robotics Laboratory.

M.I. Jordan and R.A. Jacobs. (1993). Hierarchical mixtures of experts and the EM algorithm. Technical Report 9301, MIT Computational Cognitive Science.

C. P. Lu and E. Mjolsness. (1994). Two-dimensional object localization by coarse-to-fine correlation matching. In this volume, *NIPS 6* .

C. von der Malsburg. (1988). Pattern recognition by labeled graph matching. *Neural Networks*,**1**:141:148.

E. Mjolsness and W. Miranker. (1993). Greedy Lagrangians for neural networks: three levels of optimization in relaxation dynamics. Technical Report 945, Yale University, Department of Computer Science.

E. Mjolsness. Visual grammars and their neural networks. (1992) *SPIE Conference on the Science of Artificial Neural Networks*, **1710**:63:85.

C. Peterson and B. Söderberg. A new method for mapping optimization problems onto neural networks. (1989) *International Journal of Neural Systems*,**1**(1):3:22.

P. Simard, Y. Le Cun, and J. Denker. Efficient pattern recognition using a new transformation distance. (1993). In S. Hanson, J. Cowan, and C. Giles, (eds.), *NIPS 5* . Morgan Kaufmann, San Mateo CA.